# Fiedler Random Fields: A Large-Scale Spectral Approach to Statistical Network Modeling

**Antonino Freno**          **Mikaela Keller**[*]          **Marc Tommasi**[*]

INRIA Lille – Nord Europe
40 avenue Halley – Bât A – Park Plaza
59650 Villeneuve d'Ascq (France)
{antonino.freno,mikaela.keller,marc.tommasi}@inria.fr

## Abstract

Statistical models for networks have been typically committed to strong prior assumptions concerning the form of the modeled distributions. Moreover, the vast majority of currently available models are explicitly designed for capturing some specific graph properties (such as power-law degree distributions), which makes them unsuitable for application to domains where the behavior of the target quantities is not known a priori. The key contribution of this paper is twofold. First, we introduce the Fiedler delta statistic, based on the Laplacian spectrum of graphs, which allows to dispense with any parametric assumption concerning the modeled network properties. Second, we use the defined statistic to develop the Fiedler random field model, which allows for efficient estimation of edge distributions over large-scale random networks. After analyzing the dependence structure involved in Fiedler random fields, we estimate them over several real-world networks, showing that they achieve a much higher modeling accuracy than other well-known statistical approaches.

## 1  Introduction

Arising from domains as diverse as bioinformatics and web mining, large-scale data exhibiting network structure are becoming increasingly available. Network models are commonly used to represent the relations among data units and their structural interactions. Recent studies, especially targeted at social network modeling, have focused on random graph models of those networks. In the simplest form, a random network is a configuration of binary random variables $X_{uv}$ such that the value of $X_{uv}$ stands for the presence or absence of a link between nodes $u$ and $v$ in the network. The general idea underlying random graph modeling is that network configurations are generated by a stochastic process governed by specific probability laws, so that different models correspond to different families of distributions over graphs.

The simplest random graph model is the Erdős-Rényi (ER) model [1], which assumes that the probability of observing a link between two nodes in a given graph is constant for any pair of nodes in that graph, and it is independent of which other edges are being observed. In preferential attachment models [2], the probability of linking to any specified node in a graph is proportional to the degree of the node in the graph, leading to "rich get richer" effects. Small-world models [3] try to capture instead such phenomena often observed in real networks as small diameters and high clustering coefficients. An attempt to model potentially complex dependencies between graph edges in the form of Gibbs-Boltzmann distributions is made by exponential random graph (ERG) models [4], which subsume the ER model as a special case. Finally, a recent attempt at modeling real networks through

---

[*]Université Charles de Gaulle – Lille 3, Domaine Universitaire du Pont de Bois – BP 60149, 59653 Villeneuve d'Ascq (France).

a stochastic generative process is made by Kronecker graphs [5], which try to capture phenomena such as heavy-tailed degree distributions and shrinking diameter properties while paying attention to the temporal dynamics of network growth.

While some of these models behave better than others in terms of computational tractability, one basic limitation affecting all of them is a sort of *parametric assumption* concerning the probability laws underlying the observed network properties. In other words, currently available models of network structure assume that the shape of the probability distribution generating the network is known a priori. For example, typical formulations of ERG models assume that the building blocks of real networks are given by such structures as $k$-stars and $k$-triangles, with different weights assigned to different structures, whereas the preferential attachment model is committed to the assumption that the observed degree distributions obey a power law. In such frameworks, estimating the model from data reduces to fitting the model parameters, where the parametric form of the target distribution is fixed a priori. Clearly, in order for such models to deliver accurate estimates of the distributions at hand, their prior assumptions concerning the behavior of the target quantities must be satisfied by the given data. But unfortunately, this is something that we can rarely assess a priori. To date, the knowledge we have concerning large-scale real-world networks does not allow to assess whether any particular parametric assumption is capturing in depth the target generative process, although some observed network properties may happen to be modeled fairly well.

The aim of this paper is twofold. On the one hand, we take a first step toward nonparametric modeling of random networks by developing a novel network statistic, which we call the *Fiedler delta* statistic. The Fiedler delta function allows to model different graph properties at once in an extremely compact form. This statistic is based on the spectral analysis of the graph, and in particular on the smallest non-zero eigenvalue of the Laplacian matrix, which is known as Fiedler value [6, 7]. On the other hand, we use the Fiedler delta statistic to define a Boltzmann distribution over graphs, leading to the *Fiedler random field* (FRF) model. Roughly speaking, for each binary edge variable $X_{uv}$, potentials in a FRF are functions of the difference determined in the Fiedler value by flipping the value of $X_{uv}$, where the spectral decomposition is restricted to a suitable subgraph incident to nodes $u, v$. The intuition is that the information encapsulated in the Fiedler delta for $X_{uv}$ gives a measure of the role of $X_{uv}$ in determining the algebraic connectivity of its neighborhood. As a first step in the theoretical analysis of FRFs, we prove that these models allow to capture edge correlations at any distance within a given neighborhood, hence defining a fairly general class of conditional independence structures over networks.

The paper is organized as follows. Sec. 2 reviews some theoretical background concerning the Laplacian spectrum of graphs. FRFs are then introduced in Sec. 3, where we also analyze their dependence structure and present an efficient approach for learning them from data. To avoid unwarranted prior assumptions concerning the statistical behavior of the Fiedler delta, potentials are modeled by non-linear functions, which we estimate from data by minimizing a contrastive divergence objective. FRFs are evaluated experimentally in Sec. 4, showing that they are well suited for large-scale estimation problems over real-world networks, while Sec. 5 draws some conclusions and sketches a few directions for further work.

## 2   Graphs, Laplacians, and eigenvalues

Let $\mathcal{G} = (\mathcal{V}, \mathcal{E})$ be an undirected graph with $n$ nodes. In the following we assume that the graph is unweighted with adjacency matrix $\mathbf{A}$. The degree $d_u$ of a node $u \in \mathcal{V}$ is defined as the number of connections of $u$ to other nodes, that is $d_u = |\{v : \{u, v\} \in \mathcal{E}\}|$. Accordingly, the degree matrix $\mathbf{D}$ of a graph $\mathcal{G}$ corresponds to the diagonal matrix with the vertex degrees $d_1, \ldots, d_n$ on the diagonal. The main tools exploited by the random graph model proposed here are the graph Laplacian matrices. Different graph Laplacians have been defined in the literature. In this work, we use consistently the *unnormalized graph Laplacian*, given by $\mathbf{L} = \mathbf{D} - \mathbf{A}$. Some basic facts related to the unnormalized Laplacian matrix can be summarized as follows [7]:

**Proposition 1.** *The unnormalized graph Laplacian $\mathbf{L}$ of an undirected graph $\mathcal{G}$ has the following properties: (i) $\mathbf{L}$ is symmetric and positive semi-definite; (ii) the smallest eigenvalue of $\mathbf{L}$ is 0; (iii) $\mathbf{L}$ has $n$ non-negative, real-valued eigenvalues $0 = \lambda_1 \leq \ldots \leq \lambda_n$; (iv) the multiplicity of the eigenvalue 0 of $\mathbf{L}$ equals the number of connected components in the graph, that is, $\lambda_1 = 0$ and $\lambda_2 > 0$ if and only if $\mathcal{G}$ is connected.*

In the following, the (algebraic) multiplicity of an eigenvalue $\lambda_i$ will be denoted by $M(\lambda_i, \mathcal{G})$.

If the graph has one single connected component, then $M(0, \mathcal{G}) = 1$, and the second smallest eigenvalue $\lambda_2(\mathcal{G}) > 0$ is called, in this case, the *Fiedler eigenvalue*. The Fiedler eigenvalue provides insight into several graph properties: when there is a nontrivial spectral gap, i.e. $\lambda_2(\mathcal{G})$ is clearly separated from 0, the graph has good expansion properties, stronger connectivity, and rapid convergence of random walks in the graph. For example, it is known that $\lambda_2(\mathcal{G}) \leq \mu(\mathcal{G})$, where $\mu(\mathcal{G})$ is the edge connectivity of the graph (i.e. the size of the smallest edge cut whose removal makes the graph disconnected [7]). Notice that if the graph has more than one connected component, then $\lambda_2(\mathcal{G})$ will be also equal to zero, thus implying that the graph is not connected. Without loss of generality, we abuse the term Fiedler eigenvalue to denote the smallest eigenvalue different from zero, regardless of the number of connected components. In this paper, by Fiedler value we mean the eigenvalue $\lambda_{k+1}(\mathcal{G})$, where $k = M(0, \mathcal{G})$.

For any pair of nodes $u$ and $v$ in a graph $\mathcal{G} = (\mathcal{V}, \mathcal{E})$, we define two corresponding graphs $\mathcal{G}^{uv^+}$ and $\mathcal{G}^{uv^-}$ in the following way: $\mathcal{G}^{uv^+} = (\mathcal{V}, \mathcal{E} \cup \{\{u, v\}\})$, and $\mathcal{G}^{uv^-} = (\mathcal{V}, \mathcal{E} \setminus \{\{u, v\}\})$. Clearly, we have that either $\mathcal{G}^{uv^+} = \mathcal{G}$ or $\mathcal{G}^{uv^-} = \mathcal{G}$. A basic property concerning the Laplacian eigenvalues of $\mathcal{G}^{uv^+}$ and $\mathcal{G}^{uv^-}$ is the following [7, 8, 9]:

**Lemma 1.** *If $\mathcal{G}^{uv^+}$ and $\mathcal{G}^{uv^-}$ are two graphs with $n$ nodes, such that $\{u, v\} \subseteq \mathcal{V}$, $\mathcal{G}^{uv^+} = (\mathcal{V}, \mathcal{E} \cup \{\{u, v\}\})$, and $\mathcal{G}^{uv^-} = (\mathcal{V}, \mathcal{E} \setminus \{\{u, v\}\})$, then we have that: (i) $\sum_{i=1}^{n} \lambda_i(\mathcal{G}^{uv^+}) - \lambda_i(\mathcal{G}^{uv^-}) = 2$; (ii) $\lambda_i(\mathcal{G}^{uv^+}) \leq \lambda_i(\mathcal{G}^{uv^-})$ for any $i$ such that $1 \leq i \leq n$.*

## 3  Fiedler random fields

Fiedler random fields are introduced in Sec. 3.1, while in Secs. 3.2–3.3 we discuss their dependence structure and explain how to estimate them from data respectively.

### 3.1  Probability distribution

Using the notions reviewed above, we define the Fiedler delta function $\Delta\lambda_2$ in the following way:

**Definition 1.** *Given graph $\mathcal{G}$, let $k = M(0, \mathcal{G}^{uv^+})$. Then,*

$$\Delta\lambda_2(u, v, \mathcal{G}) = \begin{cases} \lambda_{k+1}(\mathcal{G}^{uv^+}) - \lambda_{k+1}(\mathcal{G}^{uv^-}) & \text{if } X_{uv} = 1 \\ \lambda_{k+1}(\mathcal{G}^{uv^-}) - \lambda_{k+1}(\mathcal{G}^{uv^+}) & \text{otherwise} \end{cases} \tag{1}$$

In other words, $\Delta\lambda_2(u, v, \mathcal{G})$ is the variation in the Fiedler eigenvalue of the graph Laplacian that would result from flipping the value of $X_{uv}$ in $\mathcal{G}$. Concerning the range of the Fiedler delta function, we can easily prove the following proposition:

**Proposition 2.** *For any graph $\mathcal{G} = (\mathcal{V}, \mathcal{E})$ and any pair of nodes $\{u, v\}$ such that $X_{uv} = 1$, we have that $0 \leq \Delta\lambda_2(u, v, \mathcal{G}) \leq 2$.*

*Proof.* Let $k = M(0, \mathcal{G})$. The proposition follows straightforwardly from Lemma 1, given that $\Delta\lambda_2(u, v, \mathcal{G}) = \lambda_{k+1}(\mathcal{G}) - \lambda_{k+1}(\mathcal{G}^{uv^-})$. $\quad\square$

We now proceed to define FRFs. Given a graph $\mathcal{G} = (\mathcal{V}, \mathcal{E})$, for each (unordered) pair of nodes $\{u, v\}$ such that $u \neq v$, we take $X_{uv}$ to denote a binary random variable such that $X_{uv} = 1$ if $\{u, v\} \in \mathcal{E}$, and $X_{uv} = 0$ otherwise. Since the graph is undirected, $X_{uv} = X_{vu}$. We also say that a subgraph $\mathcal{G}_S$ of $\mathcal{G}$ with edge set $\mathcal{E}_S$ is *incident* to $X_{uv}$ if $\{u, v\} \subseteq \bigcup_{e \in \mathcal{E}_S} e$. Then:

**Definition 2.** *Given a graph $\mathcal{G}$, let $\boldsymbol{X}_{\mathcal{G}}$ denote the set of random variables defined on $\mathcal{G}$, i.e. $\boldsymbol{X}_{\mathcal{G}} = \{X_{uv} : u \neq v \wedge \{u, v\} \subseteq \mathcal{V}\}$. For any $X_{uv} \in \boldsymbol{X}_{\mathcal{G}}$, let $\mathcal{G}_{uv}$ be a subgraph of $\mathcal{G}$ which is incident to $X_{uv}$ and $\varphi_{uv}$ be a two-place real-valued function with parameter vector $\boldsymbol{\theta}$. We say that the probability distribution of $\boldsymbol{X}_{\mathcal{G}}$ is a* Fiedler random field *if it factorizes as*

$$P(\boldsymbol{X}_{\mathcal{G}} | \boldsymbol{\theta}) = \frac{1}{Z(\boldsymbol{\theta})} \exp\left( \sum_{X_{uv} \in \boldsymbol{X}_{\mathcal{G}}} \varphi_{uv}\big(X_{uv}, \Delta\lambda_2(u, v, \mathcal{G}_{uv}); \boldsymbol{\theta}\big) \right) \tag{2}$$

*where $Z(\boldsymbol{\theta})$ is the partition function.*

In other words, a FRF is a Gibbs-Boltzmann distribution over graphs, with potential functions defined for each node pair $\{u, v\}$ along with some neighboring subgraph $\mathcal{G}_{uv}$. In particular, in order to model the dependence of each variable $X_{uv}$ on $\mathcal{G}_{uv}$, potentials take as argument both the value of $X_{uv}$ and the Fiedler delta corresponding to $\{u, v\}$ in $\mathcal{G}_{uv}$. The idea is to treat the Fiedler delta statistic as a (real-valued) random variable defined over subgraph configurations, and to exploit this random variable as a compact representation of those configurations. This means that the dependence structure of a FRF is fixed by the particular choice of subgraphs $\mathcal{G}_{uv}$, so that the set $\boldsymbol{X}_{\mathcal{G}_{uv}} \setminus \{X_{uv}\}$ makes $X_{uv}$ independent of $\boldsymbol{X}_{\mathcal{G}} \setminus \boldsymbol{X}_{\mathcal{G}_{uv}}$. Three fundamental questions are then the following. First, how do we fix the subgraph $\mathcal{G}_{uv}$ for each pair of nodes $\{u, v\}$? Second, how do we choose a shape for the potential functions, so as to fully exploit the information contained in the Fiedler delta, while avoiding unwarranted assumptions concerning their parametric form? Third, how does the Fiedler delta statistic behave with respect to the Markov dependence property for random graphs? One basic result related to the third question is presented in Sec. 3.2, while Sec. 3.3 will address the first two points.

## 3.2 Dependence structure

We first recall the definition of Markov dependence for random graphs [10]. Let $\mathcal{N}(X_{uv})$ denote the set $\{X_{wz} \colon \{w, z\} \in \mathcal{E} \wedge |\{w, z\} \cap \{u, v\}| = 1\}$. Then:

**Definition 3.** *A random graph $\mathcal{G}$ is said to be a* Markov *graph (or to have a* Markov *dependence structure) if, for any pair of variables $X_{uv}$ and $X_{wz}$ in $\mathcal{G}$ such that $\{u, v\} \cap \{w, z\} = \emptyset$, we have that $P(X_{uv} | X_{wz}, \mathcal{N}(X_{uv})) = P(X_{uv} | \mathcal{N}(X_{uv}))$.*

Based on Def. 3, we say that the dependence structure of a random graph $\mathcal{G}$ is *non-Markovian* if, for disjoint pairs of nodes $\{u, v\}$ and $\{w, z\}$, it does not imply that $P(X_{uv} | X_{wz}, \mathcal{N}(X_{uv})) = P(X_{uv} | \mathcal{N}(X_{uv}))$, i.e. if it is consistent with the inequality $P(X_{uv} | X_{wz}, \mathcal{N}(X_{uv})) \neq P(X_{uv} | \mathcal{N}(X_{uv}))$. We can then prove the following proposition:

**Proposition 3.** *There exist Fiedler random fields with non-Markovian dependence structure.*

*Proof sketch.* Consider a graph $\mathcal{G} = (\mathcal{V}, \mathcal{E})$ such that $\mathcal{V} = \{u, v, w, z\}$ and $\mathcal{E} = \{\{u, v\}, \{v, w\}, \{w, z\}, \{u, z\}\}$. The proof relies on the following result [6]: if graphs $\mathcal{G}_1$ and $\mathcal{G}_2$ are, respectively, a path and a circuit of size $n$, then $\lambda_2(\mathcal{G}_1) = 2\left(1 - \cos(\pi/n)\right)$ and $\lambda_2(\mathcal{G}_2) = 2\left(1 - \cos(2\pi/n)\right)$. Since adding exactly one edge to a path of size 4 can yield a circuit of the same size, this property allows to derive analytic forms for the Fiedler delta statistic in such graphs, showing that there exist parameterizations of $\varphi_{uv}$ such that $\varphi_{uv}(X_{uv}, \Delta\lambda_2(u, v, \mathcal{G}); \boldsymbol{\theta}) \neq \varphi_{uv}(X_{uv}, \Delta\lambda_2(u, v, \mathcal{G}_\mathcal{S}); \boldsymbol{\theta})$. This means that the dependence structure of $\mathcal{G}$ is non-Markovian.[1] □

Note that the proof of Prop. 3 can be straightforwardly generalized to the dependence between two variables $X_{uv}$ and $X_{wz}$ in circuits/paths of arbitrary size $n$, since the expression used for the Fiedler eigenvalues of such graphs holds for any $n$. This fact suggests that FRFs allow to model edge correlations at virtually any distance within $\mathcal{G}$, provided that each subgraph $\mathcal{G}_{uv}$ is chosen in such a way as to encompass the relevant correlation.

## 3.3 Model estimation

The problem of learning a FRF from an observed network can be split into the task of estimating the potential functions once the network distribution has been factorized into a particular set of subgraphs, and the task of factorizing the distribution through a suitable set of subgraphs, which corresponds to estimating the dependence structure of the FRF. Here we focus on the problem of learning the FRF potentials, while suggesting a heuristic way to fix the dependence structure of the model.

In order to estimate the FRF potentials, we need to specify on the one hand a suitable architecture for such functions, and on the other hand the objective function that we want to optimize. As a

preliminary step, we tested experimentally a variety of shapes for the potential functions. The tests indicated the importance of avoiding limiting assumptions concerning the form of the potentials, which motivated us to model them by a feed-forward multilayer perceptron (MLP), due to its well-known capabilities of approximating functions of arbitrary shape [12]. In particular, throughout the applications described in this paper we use a simple MLP architecture with one hidden layer and hyperbolic tangent activation functions. Therefore, our parameter vector $\boldsymbol{\theta}$ simply consists of the weights of the chosen MLP architecture. Notice that, as far as the estimation of potentials is concerned, any regression model offering approximation capabilities analogous to the MLP family could be used as well. Here, the only requirement is to avoid unwarranted prior assumptions with respect to the shape of the potential functions. In this respect, we take our approach to be genuinely *nonparametric*, since it does not require the parametric form of the target functions to be specified a priori in order to estimate them accurately. Concerning instead the learning objective, the main difficulty we want to avoid is the complexity of computing the partition function involved in the Gibbs-Boltzmann distribution. The approach we adopt to this aim is to minimize a *contrastive divergence* objective [13]. If $\mathcal{G} = (\mathcal{V}, \mathcal{E})$ is the network that we want to fit our model to, and $\mathcal{G}_{uv} = (\mathcal{V}_{uv}, \mathcal{E}_{uv})$ is a subgraph of $\mathcal{G}$ such that $\{u, v\} \in \mathcal{V}_{uv}$, let $\mathcal{G}_{uv}^*$ denote the graph that we obtain by resampling the value of $X_{uv}$ in $\mathcal{G}_{uv}$ according to the conditional distribution $\widehat{P}(X_{uv} | \boldsymbol{x}_{\mathcal{G}_{uv}} \setminus \{x_{uv}\}; \boldsymbol{\theta})$ predicted by our model. In other words, $\mathcal{G}_{uv}^*$ is the result of performing just one iteration of Gibbs sampling on $X_{uv}$ using $\boldsymbol{\theta}$, where the configuration $\boldsymbol{x}_{\mathcal{G}_{uv}}$ of $\mathcal{G}_{uv}$ is used to initialize the (single-step) Markov chain. Then, our goal is to minimize the function $\ell_{CD}(\boldsymbol{\theta}; \mathcal{G})$, given by:

$$
\begin{aligned}
\ell_{CD}(\boldsymbol{\theta}; \mathcal{G}) &= \log \left\{ \frac{1}{Z(\boldsymbol{\theta})} \exp \left( \sum_{X_{uv} \in \boldsymbol{X}_{\mathcal{G}}} \varphi(x_{uv}^*, \Delta\lambda_2(u, v, \mathcal{G}_{uv}^*); \boldsymbol{\theta}) \right) \right\} - \log \widehat{P}(\boldsymbol{x}_{\mathcal{G}} | \boldsymbol{\theta}) \\
&= \sum_{X_{uv} \in \boldsymbol{X}_{\mathcal{G}}} \left\{ \varphi(x_{uv}^*, \Delta\lambda_2(u, v, \mathcal{G}_{uv}^*); \boldsymbol{\theta}) - \varphi(x_{uv}, \Delta\lambda_2(u, v, \mathcal{G}_{uv}); \boldsymbol{\theta}) \right\}
\end{aligned}
\tag{3}
$$

where $\varphi$ is the function computed by our MLP architecture. The appeal of contrastive divergence learning is that, while it does not require to compute the partition function, it is known to converge to points which are very close to maximum-likelihood solutions [14].

If we want our learning objective to be usable in the large-scale setting, then it is not feasible to sum over all node pairs $\{u, v\}$ in the network, since the number of such pairs grows quadratically with $|\mathcal{V}|$. In this respect, a straightforward approach for scaling to very large networks consists in sampling $n$ objects from the set of all possible pairs of nodes, taking care that the sample contains a good balance between linked and unlinked pairs. Another issue we need to address concerns the way we sample a suitable set of subgraphs $\mathcal{G}_{u_1 v_1}, \ldots, \mathcal{G}_{u_n v_n}$ for the selected pairs of nodes. Although different sampling techniques could be used in principle [15], our goal is to model correlations between each variable $X_{uv}$ and some neighboring region $\mathcal{G}_{uv}$ in $\mathcal{G}$. Such a neighborhood should be large enough to make $\Delta\lambda_2(u, v, \mathcal{G}_{uv})$ sufficiently informative with respect to the overall network, but also small enough to keep the spectral decomposition of $\mathcal{G}_{uv}$ computationally tractable. Therefore, in order to sample $\mathcal{G}_{uv}$, we propose to draw $\mathcal{V}_{uv}$ by performing $k$ 'snowball waves' on $\mathcal{G}$ [16], using $u$ and $v$ as seeds, and then setting $\mathcal{E}_{uv}$ to be the edge set induced by $\mathcal{V}_{uv}$ in $\mathcal{G}$ (see Algorithm 1 for the details). In this way, we can empirically tune the $k$ hyperparameter in order to trade-off the informativeness of $\mathcal{G}_{uv}$ for the tractability of its spectral decomposition, where it is known that the complexity of computing $\Delta\lambda_2(u, v, \mathcal{G}_{uv})$ is cubic with respect to the number of nodes in $\mathcal{G}_{uv}$ [17].

---

**Algorithm 1** `SampleSubgraph`: Sampling a neighboring subgraph for a given node pair

---

**Input:** Undirected graph $\mathcal{G} = (\mathcal{V}, \mathcal{E})$; node pair $\{u, v\}$; number $k$ of snowball waves.
**Output:** Undirected graph $\mathcal{G}_{uv} = (\mathcal{V}_{uv}, \mathcal{E}_{uv})$.

```
SampleSubgraph(G, {u, v}, k):
1.   V_uv = {u, v}
2.   for(i = 1 to k)
3.      V_uv = V_uv ∪ ⋃_{w∈V_uv}{z ∈ V: {w, z} ∈ E}
4.   E_uv = {{w, z} ∈ E: {w, z} ⊆ V_uv}
5.   return (V_uv, E_uv)
```

---

Once sampled our training set $\mathcal{D} = \big\{(x_{u_1v_1}, \mathcal{G}_{u_1v_1}), \dots, (x_{u_nv_n}, \mathcal{G}_{u_nv_n})\big\}$, we learn the MLP weights by minimizing the objective $\ell_{CD}(\boldsymbol{\theta}; \mathcal{D})$, which which we obtain from $\ell_{CD}(\boldsymbol{\theta}; \mathcal{G})$ by restricting the summation in Eq. 3 to the elements of $\mathcal{D}$. Minimization is performed by iterative gradient descent, using standard backpropagation for updating the MLP weights.

## 4 Experimental evaluation

In order to investigate the empirical behavior of FRFs as models of large-scale networks, we design two different groups of experiments (in link prediction and graph generation respectively), using collaboration networks drawn from the arXiv e-print repository (`http://snap.stanford.edu/data/index.html`), where nodes represent scientists and edges represent paper coauthorships. Some basic network statistics are reported in Table 1.

**Link prediction.** In the first kind of experiments, given a random network $\mathcal{G} = (\mathcal{V}, \mathcal{E})$, our goal is to measure the accuracy of FRFs at estimating the conditional distribution of variables $X_{uv}$ given the configuration of neighboring subgraphs $\mathcal{G}_{uv}$ of $\mathcal{G}$. This can be seen as a link prediction problem where only local information (given by $\mathcal{G}_{uv}$) can be used for predicting the presence of a link $\{u, v\}$. At the same time, we want to understand whether the overall network size (in terms of the number of nodes) has an impact on the number of training examples that will be necessary for FRFs to converge to stable prediction accuracy. Recall that FRFs are trained on a data sample $\mathcal{D} = \big\{(x_{u_1v_1}, \mathcal{G}_{u_1v_1}), \dots, (x_{u_nv_n}, \mathcal{G}_{u_nv_n})\big\}$, where $n \ll \frac{|\mathcal{V}|(|\mathcal{V}|-1)}{2}$. Given this, converging to stable predictions for values of $n$ which do not depend on $|\mathcal{V}|$ is a crucial requirement for achieving large-scale applicability. Let us sample our training set $\mathcal{D}$ by first drawing $n$ node pairs from $\mathcal{V}$ in such a way that linked and unlinked pairs from $\mathcal{G}$ are equally represented in $\mathcal{D}$, and then extracting the corresponding subgraphs $\mathcal{G}_{u_i, v_i}$ by Algorithm 1 using one snowball wave. We then learn our model from $\mathcal{D}$ as described in Sec. 3.3. In all the experiments reported in this work, the number of hidden units in our MLP architecture is set to 5. A test set $\mathcal{T}$ containing $m$ objects $(x_{u_1v_1}, \mathcal{G}_{\mathcal{S}_1}), \dots, (x_{u_mv_m}, \mathcal{G}_{\mathcal{S}_m})$ is also sampled from $\mathcal{G}$ so that $\mathcal{T} \cap \mathcal{D} = \emptyset$, where pairs $\{u_i, v_i\}$ in $\mathcal{T}$ are drawn uniformly at random from $\mathcal{V} \times \mathcal{V}$.

Predictions are derived from the learned model by first computing the conditional probability of observing a link for each pair of nodes $\{u_j, v_j\}$ in $\mathcal{T}$, and then making a decision on the presence/absence of links by thresholding the predicted probability (where the threshold is tuned by cross-validation). Prediction accuracy is measured by averaging the recognition accuracy for linked and unlinked pairs in $\mathcal{T}$ respectively (where $|\mathcal{T}| = 10,000$). In Fig. 1, the accuracy of FRFs on the test set is plotted against a growing size $n$ of $\mathcal{D}$ (where $12 \leq n \leq 48$). Interestingly, the number of training examples required for the accuracy curve to stabilize does not seem to depend at all on the overall network size. Indeed, fastest convergence is achieved for the average-sized and the second largest networks, i.e. HepPh and AstroPh respectively.

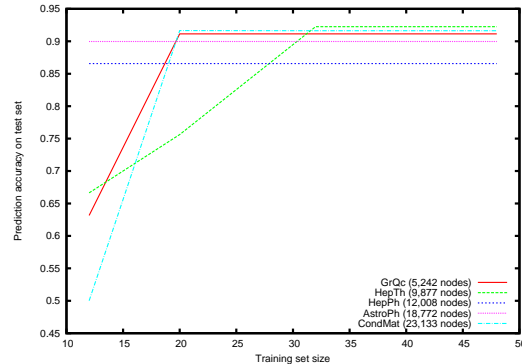

Figure 1: Prediction accuracy of FRFs on the arXiv networks for a growing training set size.

Notice how a training sample containing an extremely small percentage of node pairs is sufficient for our learning approach to converge to stable prediction accuracy. This result encourages to think of FRFs as a convenient modeling option for the large-scale setting.

Besides assessing whether the network size affects the number of training samples needed to accurately learn FRFs, we want to evaluate the usefulness of the dependence structure involved in our model in predicting the conditional distributions of edges given their neighboring subgraphs. That is, we want to ascertain whether the effort of modeling the conditional independence structure of the overall network through the FRF formalism is justified by a suitable gain in prediction accuracy with respect to statistical models that do not focus explicitly on such dependence structure. To this aim, we compare FRFs to two popular statistical models for large-scale networks, namely the Watts-Strogatz (WS) and the Barabási-Albert (BA) models [3, 2]. The WS formalism is mainly aimed

at modeling the short-diameter property often observed in real-world networks. Interestingly, the degree distribution of WS networks can be expressed in closed form in terms of two parameters $\delta$ and $\beta$, related to the average degree distribution and a network rewiring process respectively [18]. On the other hand, the BA model is aimed at explaining the emergence of power-law degree distributions, where such distributions can be expressed in terms of an adaptive parameter $\alpha$ [19]. The parameters of both the WS and the BA model can be estimated by standard maximum-likelihood approaches and then used to predict conditional edge distributions, exploiting information from the degrees observed in the given subgraphs [20, 21]. The ER model is not considered in this group of experiments, since the involved independence assumption makes it unusable (i.e. equivalent to random guessing) for the purposes of conditional estimation tasks. On the other hand, ERG models are not suitable for application to the large-scale setting. We tried them out using edge, $k$-star and $k$-triangle statistics [4], and the tests confirmed this point. Although the prohibitive cost of fitting the models and computing the involved feature functions could be overcome in principle by sampling strategies similar to the ones we employ for FRFs, the potentials used in ERGs become numerically unstable in the large-scale setting, leading to numerical representation issues for which we are not aware of any off-the-shelf solution. Accuracy values for the different models are reported in Table 1. FRFs dramatically outperform the other two models on all networks. Since both the BA and the WS model do not show relevant improvements over simple random guessing, this result clearly suggests that exploiting the dependence structure involved in network edge configurations is crucial to accurately predict the presence/absence of links.

Table 1: Edge prediction results on the arXiv networks. General network statistics are also reported, where $CC_\mathcal{G}$ and $D_\mathcal{G}$ stand for average clustering coefficient and network diameter respectively.

| Dataset | Network Statistics | | | | Prediction Accuracy | | |
|---|---|---|---|---|---|---|---|
| | $|\mathcal{V}|$ | $|\mathcal{E}|$ | $CC_\mathcal{G}$ | $D_\mathcal{G}$ | BA | FRF | WS |
| **AstroPh** | 18,772 | 396,160 | 0.63 | 14 | 50.98% | **89.97%** | 50.14% |
| **CondMat** | 23,133 | 186,936 | 0.63 | 15 | 50.15% | **91.62%** | 56.71% |
| **GrQc** | 5,242 | 28,980 | 0.52 | 17 | 52.57% | **91.14%** | 53.72% |
| **HepPh** | 12,008 | 237,010 | 0.61 | 13 | 51.61% | **86.57%** | 54.33% |
| **HepTh** | 9,877 | 51,971 | 0.47 | 17 | 58.33% | **92.25%** | 50.30% |

**Graph generation.** A second group of experiments is aimed at assessing whether the FRFs learned on the arXiv networks can be considered as plausible models of the degree distribution (DD) and the clustering coefficient distribution (CC) observed in each network [15]. To this aim, we use the estimated FRF models to generate artificial graphs of various size, using Gibbs sampling, and then we compare the DD and CC observed in the artificial graphs with those estimated on the whole networks. For scale-free networks such as the ones considered here, the BA model is known to be the most accurate model currently available with respect to DD. On the other hand, for CC both BA and WS are known to be more realistic models than ER random graphs. Therefore, we compare the graphs generated by FRFs to those generated by the BA, ER, and WS models for the same networks. The distance in DD and CC between the artificial graphs on the one hand and the corresponding real network on the other hand is measured using the Kolmogorov-Smirnov $D$-statistic, following a common use in graph mining research [15]. Here we only plot results for the CondMat and HepTh networks, noticing that the results we collected on the other arXiv networks lend themselves to the same interpretation as the ones displayed in Fig. 2. Values are averaged over 100 samples for each considered graph size, where the standard deviation is typically in the order of $10^{-2}$. The outcome motivates the following considerations. Concerning DD, FRFs are able to improve (at least slightly) the accuracy of the state-of-the-art BA model, while they are very close that model with respect to clustering coefficient. In all cases, both BA and FRFs prove to be far more accurate than ER or WS, where the only advantage of using WS is limited to improving CC over ER. These results are particularly encouraging, since they show how the nonparametric approach motivating the FRF model allows to accurately estimate network properties (such as DD) that are not aimed for explicitly in the model design. This suggests that the Fiedler delta statistic is a promising direction for building generative models capable of capturing different network properties through a unified approach.

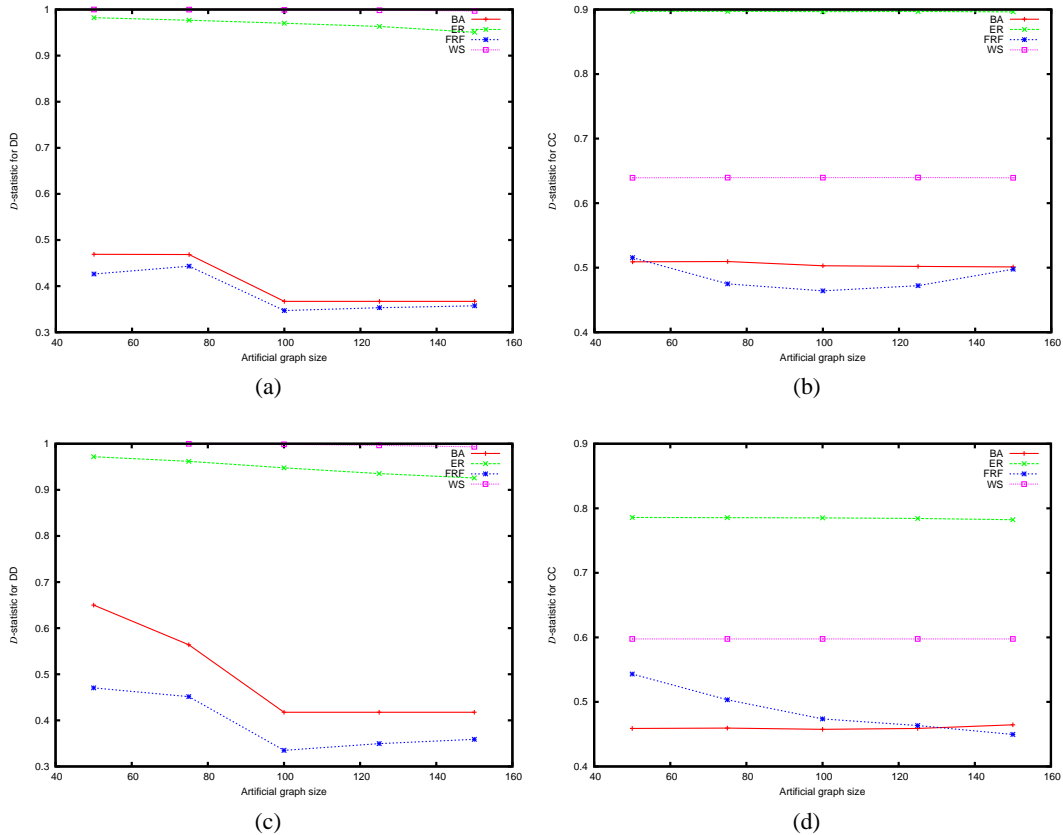

Figure 2: $D$-statistic values for DD and CC on the CondMat (a–b) and HepTh (c–d) networks.

## 5   Conclusions and future work

The main motivation inspiring this work was the observation that statistical modeling of networks cries for genuinely nonparametric estimation, because of the inaccuracy often resulting from unwarranted parametric assumptions. In this respect, we showed how the Fiedler delta statistic offers a powerful building block for designing a nonparametric estimator, which we developed in the form of the FRF model. Since here we only applied FRFs to collaboration networks, which are typically scale-free, an important option for future work is to assess the flexibility of FRFs in modeling networks from different families. In the second place, since we only addressed in a heuristic way the problem of learning the dependence structure of FRFs, a stimulating direction for further research consists in designing clever techniques for learning the structure of FRFs, e.g. considering the use of alternative subgraph sampling techniques. Finally, we would like to assess the possibility of modeling networks through mixtures of FRFs, so as to fit different network regions (with possibly conflicting properties) through specialized components of the mixture.

#### Acknowledgments

This work has been supported by the French National Research Agency (ANR-09-EMER-007). The authors are grateful to Gemma Garriga, Rémi Gilleron, Liva Ralaivola, and Michal Valko for their useful suggestions and comments.

## Footnotes

[1]For a complete proof, see the supplementary material.

## References

[1] P. Erdős and A. Rényi, "On Random Graphs, I," *Publicationes Mathematicae Debrecen*, vol. 6, pp. 290–297, 1959.

[2] A.-L. Barabási and R. Albert, "Emergence of scaling in random networks," *Science*, vol. 286, pp. 509–512, 1999.

[3] D. J. Watts and S. H. Strogatz, "Collective dynamics of 'small-world' networks," *Nature*, vol. 393, pp. 440–442, 1998.

[4] T. A. B. Snijders, P. E. Pattison, G. L. Robins, and M. S. Handcock, "New Specifications for Exponential Random Graph Models," *Sociological Methodology*, vol. 36, pp. 99–153, 2006.

[5] J. Leskovec, D. Chakrabarti, J. Kleinberg, C. Faloutsos, and Z. Ghahramani, "Kronecker graphs: An approach to modeling networks," *Journal of Machine Learning Research*, vol. 11, pp. 985–1042, 2010.

[6] M. Fiedler, "Algebraic connectivity of graphs," *Czechoslovak Mathematical Journal*, vol. 23, pp. 298–305, 1973.

[7] B. Mohar, "The Laplacian Spectrum of Graphs," in *Graph Theory, Combinatorics, and Applications* (Y. Alavi, G. Chartrand, O. R. Oellermann, and A. J. Schwenk, eds.), pp. 871–898, Wiley, 1991.

[8] W. N. Anderson and T. D. Morley, "Eigenvalues of the Laplacian of a graph," *Linear and Multilinear Algebra*, vol. 18, pp. 141–145, 1985.

[9] D. M. Cvetković, M. Doob, and H. Sachs, eds., *Spectra of Graphs: Theory and Application*. New York (NY): Academic Press, 1979.

[10] O. Frank and D. Strauss, "Markov Graphs," *Journal of the American Statistical Association*, vol. 81, pp. 832–842, 1986.

[11] J. Besag, "Spatial Interaction and the Statistical Analysis of Lattice Systems," *Journal of the Royal Statistical Society. Series B*, vol. 36, pp. 192–236, 1974.

[12] K. Hornik, "Approximation capabilities of multilayer feedforward networks," *Neural Networks*, vol. 4, no. 2, pp. 251–257, 1991.

[13] G. E. Hinton, "Training Products of Experts by Minimizing Contrastive Divergence," *Neural Computation*, vol. 14, no. 8, pp. 1771–1800, 2002.

[14] M. Á. Carreira-Perpiñán and G. E. Hinton, "On Contrastive Divergence Learning," in *Proceedings of the Tenth International Workshop on Articial Intelligence and Statistics (AISTATS 2005)*, pp. 33–40, 2005.

[15] J. Leskovec and C. Faloutsos, "Sampling from large graphs," in *Proceedings of the Twelfth ACM SIGKDD International Conference on Knowledge Discovery and Data Mining (KDD 2006)*, pp. 631–636, 2006.

[16] E. D. Kolaczyk, *Statistical Analysis of Network Data. Methods and Models*. New York (NY): Springer, 2009.

[17] Z. Bai, J. Demmel, J. Dongarra, A. Ruhe, and H. van der Vorst, eds., *Templates for the Solution of Algebraic Eigenvalue Problems: A Practical Guide*. Philadelphia (PA): SIAM, 2000.

[18] A. Barrat and M. Weigt, "On the properties of small-world network models," *The European Physical Journal B*, vol. 13, pp. 547–560, 2000.

[19] R. Albert and A.-L. Barabási, "Statistical mechanics of complex networks," *Reviews of Modern Physics*, vol. 74, pp. 47–97, 2002.

[20] M. E. J. Newman, "Clustering and preferential attachment in growing networks," *Physical Review E*, vol. 64, p. 025102, 2001.

[21] A. Barabási, H. Jeong, Z. Néda, E. Ravasz, A. Schubert, and T. Vicsek, "Evolution of the social network of scientic collaborations," *Physica A*, vol. 311, pp. 590–614, 2002.

